# Clusters and Coarse Partitions in LP Relaxations

**David Sontag**
CSAIL, MIT
dsontag@csail.mit.edu

**Amir Globerson**
School of Computer Science and Engineering
The Hebrew University
gamir@cs.huji.ac.il

**Tommi Jaakkola**
CSAIL, MIT
tommi@csail.mit.edu

## Abstract

We propose a new class of consistency constraints for Linear Programming (LP) relaxations for finding the most probable (MAP) configuration in graphical models. Usual cluster-based LP relaxations enforce joint consistency on the beliefs of a cluster of variables, with computational cost increasing exponentially with the size of the clusters. By *partitioning* the state space of a cluster and enforcing consistency only across partitions, we obtain a class of constraints which, although less tight, are computationally feasible for large clusters. We show how to solve the cluster selection and partitioning problem monotonically in the dual LP, using the current beliefs to guide these choices. We obtain a dual message passing algorithm and apply it to protein design problems where the variables have large state spaces and the usual cluster-based relaxations are very costly. The resulting method solves many of these problems exactly, and significantly faster than a method that does not use partitioning.

## 1 Introduction

A common inference task in graphical models is finding the most likely setting of the values of the variables (the MAP assignment). Indeed, many important practical problems can be formulated as MAP problems (e.g., protein-design problems [9]). The complexity of the MAP problem depends on the structure of the dependencies between the variables (i.e. the graph structure) and is known to be NP-hard in general. Specifically, for problems such as protein-design, the underlying interaction graphs are dense, rendering standard exact inference algorithms useless.

A great deal of effort has been spent recently on developing approximate algorithms for the MAP problem. One promising approach is based on linear programming relaxations, solved via message passing algorithms akin to belief propagation [2, 3]. In this case, the MAP problem is first cast as an integer linear program, and then is relaxed to a linear program by removing the integer constraints and adding new constraints on the continuous variables. Whenever the relaxed solution is integral, it is guaranteed to be the optimal solution. However, this happens only if the relaxation is sufficiently "tight" (with respect to a particular objective function).

Relaxations can be made increasingly tight by introducing LP variables that correspond to clusters of variables in the original model. In fact, in recent work [6] we have shown that by adding a set of clusters over three variables, complex problems such as protein-design and stereo-vision may be solved exactly. The problem with adding clusters over variables is that computational cost scales exponentially with the cluster size. Consider, for example, a problem where each variable has 100 states (cf. protein-design). Using clusters of $s$ variables means adding $100^s$ LP variables, which is computationally demanding even for clusters of size three.

Our goal in the current paper is to design methods that introduce constraints over clusters at a reduced computational cost. We achieve this by representing clusters at a coarser level of granularity. The key observation is that it may not be necessary to represent all the possible joint states of a cluster of variables. Instead, we partition the cluster's assignments at a coarser level, and enforce consistency

only across such partitions. This removes the number of states per variable from consideration, and instead focuses on resolving currently ambiguous settings of the variables. Following the approach of [2], we formulate a dual LP for the partition-based LP relaxations and derive a message passing algorithm for optimizing the dual LP based on block coordinate descent. Unlike standard message passing algorithms, the algorithm we derive involves passing messages between coarse and fine representations of the same set of variables.

**MAP and its LP relaxation.** We consider discrete pairwise Markov random fields on a graph $G = (V, E)$, defined as the following exponential family distribution[1]

$$p(\boldsymbol{x}; \boldsymbol{\theta}) = \frac{1}{Z} e^{\sum_{ij \in E} \theta_{ij}(x_i, x_j)} \tag{1}$$

Here $\boldsymbol{\theta}$ is a parameter vector specifying how pairs of variables in $E$ interact. The MAP problem we consider here is to find the most likely assignment of the variables under $p(\boldsymbol{x}; \boldsymbol{\theta})$ (we assume that the evidence has already been incorporated into the model). This is equivalent to finding the assignment $\boldsymbol{x}_M$ that maximizes the function $f(\boldsymbol{x}; \boldsymbol{\theta}) = \sum_{ij \in E} \theta_{ij}(x_i, x_j)$.

The resulting discrete optimization problem may also be cast as a linear program. Define $\boldsymbol{\mu}$ to be a vector of marginal probabilities associated with the interacting pairs of variables (edges) $\{\mu_{ij}(x_i, x_j)\}_{ij \in E}$ as well as $\{\mu_i(x_i)\}_{i \in V}$ for the nodes. The set of $\boldsymbol{\mu}$'s that could arise from some joint distribution on $G$ is known as the *marginal polytope* $\mathcal{M}(G)$ [7]. The MAP problem is then equivalent to the following linear program:

$$\max_{\boldsymbol{x}} f(\boldsymbol{x}; \boldsymbol{\theta}) = \max_{\boldsymbol{\mu} \in \mathcal{M}(G)} \boldsymbol{\mu} \cdot \boldsymbol{\theta} , \tag{2}$$

where $\boldsymbol{\mu} \cdot \boldsymbol{\theta} = \sum_{ij \in E} \sum_{x_i, x_j} \theta_{ij}(x_i, x_j) \mu_{ij}(x_i, x_j)$. The extreme points of the marginal polytope are integral and correspond one-to-one with assignments $\boldsymbol{x}$. Thus, there always exists a maximizing $\boldsymbol{\mu}$ that is integral and corresponds to $\boldsymbol{x}_M$. Although the number of variables in this LP is only $O(|E| + |V|)$, the difficulty comes from an exponential number of linear inequalities typically required to describe the marginal polytope $\mathcal{M}(G)$.

LP relaxations replace the difficult global constraint that the marginals in $\boldsymbol{\mu}$ must arise from some common joint distribution by ensuring only that the marginals are locally consistent with one another. The most common such relaxation, *pairwise consistency*, enforces that the edge marginals are consistent with the node marginals, $\{\boldsymbol{\mu} \mid \sum_{x_j} \mu_{ij}(x_i, x_j) = \mu_i(x_i)\}$. The integral extreme points of this *local marginal polytope* also correspond to assignments. If a solution is obtained at one such extreme point, it is provably the MAP assignment. However, the local marginal polytope also contains fractional extreme points, and, as a relaxation, will in general not be tight.

We are therefore interested in tightening the relaxation. There are many known ways to do so, including cycle inequalities [5] and semi-definite constraints [8]. However, perhaps the most straightforward approach corresponds to *lifting* the relaxation by adding marginals over clusters of nodes to the model (cf. generalized belief propagation [10]) and constraining them to be consistent with the edge marginals. However, each cluster comes with a computational cost that grows as $k^s$, where $s$ is the number of variables in the cluster and $k$ is the number of states for each variable. We seek to offset this exponential cost by introducing coarsened clusters, as we show next.

## 2    Coarsened clusters and consistency constraints

We begin with an illustrative example. Suppose we have a graphical model that is a triangle with each variable taking $k$ states. We can recover the exact marginal polytope in this case by forcing the pairwise marginals $\mu_{ij}(x_i, x_j)$ to be consistent with some distribution $\mu_{123}(x_1, x_2, x_3)$. However, when $k$ is large, introducing the corresponding $k^3$ variables to our LP may be too costly and perhaps unnecessary, if a weaker consistency constraint would already lead to an integral extreme point. To this end, we will use a coarse-grained version of $\mu_{123}$ where the joint states are partitioned into larger collections, and consistency is enforced over the partitions.

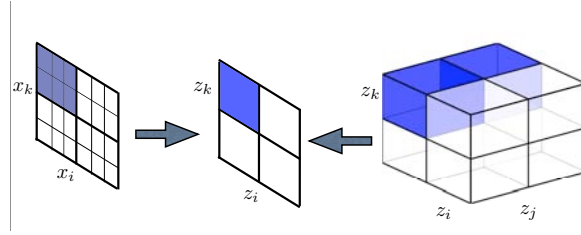

Figure 1: A graphical illustration of the consistency constraint between the original (fine granularity) edge $(x_i, x_k)$ and the coarsened triplet $(z_i, z_j, z_k)$. The two should agree on the marginal of $z_i, z_k$. For example, the shaded area in all three figures represents the same probability mass.

The simplest partitioning scheme builds on coarse-grained versions of each variable $X_i$. Let $Z_i$ denote a disjoint collection of sets covering the possible values of $X_i$. For example, if variable $X_i$ has five states, $Z_i$ might be defined as $\{1,2\}\{3,5\}\{4\}$. Given such a partitioning scheme, we can introduce a distribution over coarsened variables $\tau_{123}(z_1, z_2, z_3)$ and constrain it to agree with $\tau_{ik}(x_i, x_k)$ in the sense that they both yield the same marginals for $z_i, z_k$. This is illustrated graphically in Fig. 1. In the case when $Z_i$ individuates each state, i.e., $\{1\}\{2\}\{3\}\{4\}$, we recover the usual cluster consistency constraint.

We use the above idea to construct tighter outer bounds on the marginal polytope and incorporate them into the MAP-LP relaxation. We assume that we are given a set of clusters $\mathcal{C}$. For each cluster $c \in C$ and variable $i \in c$ we also have a partition $Z_i^c$ as in the above example[2] (the choice of clusters and partitions will be discussed later). We introduce marginals over the coarsened clusters $\tau_c(z_c)$ and constrain them to agree with the edge variables $\tau_{ij}(x_i, x_j)$ for all edges $ij \in c$:

$$\sum_{x_i \in z_i^c, x_j \in z_j^c} \tau_{ij}(x_i, x_j) = \sum_{z_c \sim z_i^c, z_j^c} \tau_c(z_c) \qquad (3)$$

The key idea is that the coarsened cluster represents higher-order marginals albeit at a lower resolution, whereas the edge variables represent lower-order marginals but at a finer resolution. The constraint in Eq. 3 implies that these two representations should agree.

We can now state the LP that we set out to solve. Our LP optimizes over the following marginal variables: $\tau_{ij}(x_i, x_j), \tau_i(x_i)$ for the edges and nodes of the original graph, and $\tau_c(z_c)$ for the coarse-grained clusters. We would like to constrain these variables to belong to the following outer bound on the marginal polytope:

$$\mathcal{M}_{\mathcal{C}}(G) = \left\{ \tau \geq 0 \;\middle|\; \begin{array}{rcl} \sum_{x_j} \tau_{ij}(x_i, x_j) &=& \tau_i(x_i) \\ \sum_{x_i \in z_i^c, x_j \in z_j^c} \tau_{ij}(x_i, x_j) &=& \sum_{z_c \sim z_i^c, z_j^c} \tau_c(z_c) \\ \sum_{x_i, x_j} \tau_{ij}(x_i, x_j) &=& 1 \end{array} \right\} \qquad (4)$$

Note that $\sum_{z_c} \tau_c(z_c) = 1$ is implied by the above constraints. The corresponding MAP-LP relaxation is then:

$$\max_{\mathcal{M}_{\mathcal{C}}(G)} \qquad (5)$$

This LP could in principle be solved using generic LP optimization tools. However, a more efficient and scalable approach is to solve it via message passing in the dual LP, which we show how to do in the next section. In addition, for this method to be successful, it is critical that we choose *good* coarsenings, meaning that it should have few partitions per variable, yet still sufficiently tightens the relaxation. Our approach for choosing the coarsenings is to iteratively solve the LP using an initial relaxation (beginning with the pairwise consistency constraints), then to introduce additional cluster constraints, letting the current solution guide how to coarsen the variables. As we showed in earlier work [6], solving with the dual LP gives us a simple method for "warm starting" the new LP (the tighter relaxation) using the previous solution, and also results in an algorithm for which every step monotonically decreases an upper bound on the MAP assignment. We will give further details of the coarsening scheme in Section 4.

# 3 Dual linear program and a message passing algorithm

In this section we give the dual of the partition-based LP from Eq. 5, and use it to obtain a message passing algorithm to efficiently optimize this relaxation. Our approach extends earlier work by Globerson and Jaakkola [2] who gave the generalized max-product linear programming (MPLP) algorithm to solve the usual (non-coarsened) cluster LP relaxation in the dual.

The dual formulation in [2] was derived by adding auxiliary variables to the primal. We followed a similar approach to obtain the LP dual of Eq. 5. The dual variables are as follows: $\beta_{ij\to i}(x_i,x_j), \beta_{ij\to j}(x_i,x_j), \beta_{ij\to ij}(x_i,x_j)$ for every edge $ij \in E$, and $\beta_{c\to ij}(z_c)$ for every coarsened cluster $c$ and edge $ij \in c$. As in [2], we define the following functions of $\beta$:

$$\lambda_{ij\to i}(x_i) \;=\; \max_{x_j}\beta_{ij\to i}(x_i,x_j), \qquad \lambda_{ij\to ij}(x_i,x_j) \;=\; \beta_{ij\to ij}(x_i,x_j) \qquad (6)$$

$$\lambda_{c\to ij}(z_i^c,z_j^c) \;=\; \max_{z_c\setminus\{z_i^c,z_j^c\}} \beta_{c\to ij}(z_c) \qquad (7)$$

As we show below, the variables $\lambda$ correspond to the messages sent in the message passing algorithm that we use for optimizing the dual. Thus $\lambda_{ij\to i}(x_i)$ should be read as the message sent from edge $ij$ to node $i$, and $\lambda_{c\to ij}(z_i^c,z_j^c)$ is the message from the coarsened cluster to one of its intersection edges. Finally, $\lambda_{ij\to ij}(x_i,x_j)$ is the message sent from an edge to itself. The dual of Eq. 5 is the following constrained minimization problem:

$$\min_{\beta} \quad \sum_i \max_{x_i} \sum_{k\in N(i)} \lambda_{ik\to i}(x_i) + \sum_{ij\in E} \max_{x_i,x_j}\Big[\lambda_{ij\to ij}(x_i,x_j) + \sum_{c:ij\in c}\lambda_{c\to ij}(z_i^c[x_i],z_j^c[x_j])\Big]$$

$$\text{s.t.} \quad \beta_{ij\to i}(x_i,x_j) + \beta_{ij\to j}(x_i,x_j) + \beta_{ij\to ij}(x_i,x_j) = \theta_{ij}(x_i,x_j) \qquad \forall ij\in E, x_i, x_j$$

$$\sum_{ij\in c}\beta_{c\to ij}(z_c) = 0 \qquad \forall c, z_c \qquad (8)$$

The notation $z_i^c[x_i]$ refers to the mapping from $x_i \in X_i$ to the coarse state $z_i^c \in Z_i^c$ such that $x_i \in z_i^c$. By convex duality, the dual objective evaluated at a dual feasible point upper bounds the primal LP optimum, which in turn upper bounds the value of the MAP assignment. It is illustrative to compare this dual LP with [2] where the cluster dual variables were $\beta_{c\to ij}(x_c)$. Our dual corresponds to introducing the additional constraint that $\beta_{c\to ij}(x_c) = \beta_{c\to ij}(x_c')$ whenever $z_c[x_c] = z_c[x_c']$.

The advantage of the above dual is that it can be optimized via a simple message passing algorithm that corresponds to block coordinate descent. The key idea is that it is possible to fix the values of the $\beta$ variables corresponding to all clusters except one, and to find a closed form solution for the non-fixed $\beta$s. It then turns out that one does not need to work with $\beta$ variables directly, but can keep only the $\lambda$ *message* variables. Fig. 2 provides the form of the updates for all three message types. $\mathcal{S}(c)$ is the set of edges in cluster $c$ (e.g. $ij, jk, ik$). Importantly, all messages outgoing from a cluster or edge must be sent simultaneously.

Here we derive the cluster to edge updates, which differ from [2]. Assume that all values of $\beta$ are fixed except for $\beta_{c\to ij}(z_i^c,z_j^c)$ for all $ij \in c$ in some cluster $c$. The term in the dual objective that depends on $\beta_{c\to ij}(z_i^c,z_j^c)$ can be written equivalently as

$$\max_{x_i,x_j}\Big[\lambda_{ij\to ij}(x_i,x_j) + \sum_{c':c'\neq c,ij\in c'}\lambda_{c'\to ij}(z_i^{c'}[x_i],z_j^{c'}[x_j]) + \lambda_{c\to ij}(z_i^c[x_i],z_j^c[x_j])\Big]$$

$$= \max_{z_i^c,z_j^c}\Big[b_{ij}(z_i^c,z_j^c) + \lambda_{c\to ij}(z_i^c[x_i],z_j^c[x_j])\Big]. \qquad (11)$$

Due to the constraint $\sum_{ij\in c}\beta_{c\to ij}(z_c) = 0$, all of the $\beta_{c\to ij}$ need to be updated simultaneously. It can be easily shown (using an equalization argument as in [2]) that the $\beta_{c\to ij}(z_c)$ that satisfy the constraint *and* minimize the objective are given by

$$\beta_{c\to ij}(z_c) = -b_{ij}(z_i^c,z_j^c) + \frac{1}{|\mathcal{S}(c)|}\sum_{st\in c}b_{st}(z_s^c,z_t^c). \qquad (12)$$

The message update given in Fig. 2 follows from the definition of $\lambda_{c\to ij}$. Note that none of the cluster messages involve the original cluster variables $x_c$, but rather only $z_c$. Thus, we have achieved the goal of both representing higher-order clusters and doing so at a reduced computational cost.

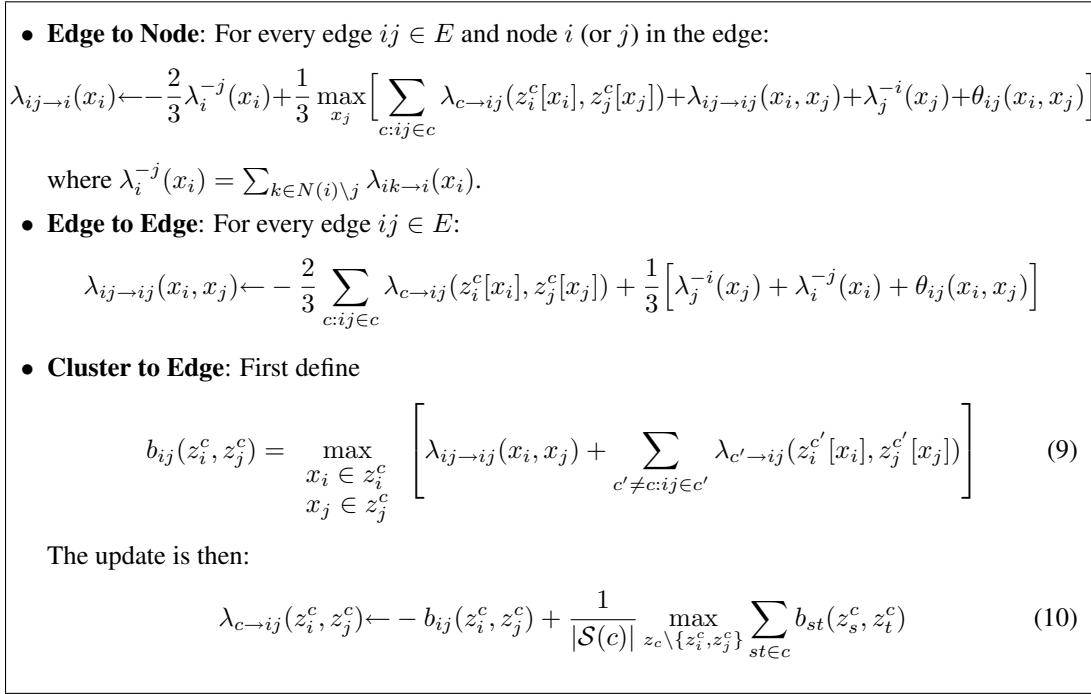

- **Edge to Node**: For every edge $ij \in E$ and node $i$ (or $j$) in the edge:

$$\lambda_{ij \to i}(x_i) \leftarrow -\frac{2}{3}\lambda_i^{-j}(x_i) + \frac{1}{3}\max_{x_j}\Big[\sum_{c:ij \in c}\lambda_{c \to ij}(z_i^c[x_i], z_j^c[x_j]) + \lambda_{ij \to ij}(x_i, x_j) + \lambda_j^{-i}(x_j) + \theta_{ij}(x_i, x_j)\Big]$$

 where $\lambda_i^{-j}(x_i) = \sum_{k \in N(i)\setminus j}\lambda_{ik \to i}(x_i)$.

- **Edge to Edge**: For every edge $ij \in E$:

$$\lambda_{ij \to ij}(x_i, x_j) \leftarrow -\frac{2}{3}\sum_{c:ij \in c}\lambda_{c \to ij}(z_i^c[x_i], z_j^c[x_j]) + \frac{1}{3}\Big[\lambda_j^{-i}(x_j) + \lambda_i^{-j}(x_i) + \theta_{ij}(x_i, x_j)\Big]$$

- **Cluster to Edge**: First define

$$b_{ij}(z_i^c, z_j^c) = \max_{\substack{x_i \in z_i^c \\ x_j \in z_j^c}}\left[\lambda_{ij \to ij}(x_i, x_j) + \sum_{c' \neq c:ij \in c'}\lambda_{c' \to ij}(z_i^{c'}[x_i], z_j^{c'}[x_j])\right] \qquad (9)$$

 The update is then:

$$\lambda_{c \to ij}(z_i^c, z_j^c) \leftarrow -b_{ij}(z_i^c, z_j^c) + \frac{1}{|\mathcal{S}(c)|}\max_{z_c \setminus \{z_i^c, z_j^c\}}\sum_{st \in c}b_{st}(z_s^c, z_t^c) \qquad (10)$$

Figure 2: The message passing updates for solving the dual LP given in Eq. 8.

The algorithm in Fig. 2 solves the dual for a given choice of coarsened clusters. As mentioned in Sec. 2, we would like to add such clusters gradually, as in [6]. Our overall algorithm is thus similar in structure to [6] and proceeds as follows (we denote the message passing algorithm from Fig. 2 by MPLP): **1.** Run MPLP until convergence using the pairwise relaxation, **2.** Find an integral solution $\boldsymbol{x}$ by locally maximizing the single node beliefs $b_i(x_i) = \sum_{k \in N(i)}\lambda_{ki \to i}(x_i)$, **3.** If the dual objective given in Eq. 8 is sufficiently close to the primal objective $f(\boldsymbol{x}; \boldsymbol{\theta})$, terminate, **4.** Add a new coarsened cluster $c$ using the strategy given in Sec. 4, **5.** Initialize messages going out of the new cluster $c$ to zero, and keep all the previous message values (this will not change the bound value), **6.** Run MPLP for $N$ iterations, then return to **2.**

## 4  Choosing coarse partitions

Until now we have not discussed how to choose the clusters to add and their partitionings. Our strategy for doing so closely follows that of our earlier work [6]. Given a set $\mathcal{C}$ of candidate clusters to add (e.g., the set of all triplets in the graph as in [6]), we would like to add a cluster that would result in the maximum decrease of the dual bound on the MAP. In principle such a cluster could be found by optimizing the dual for each candidate cluster, then choosing the best one. However, this is computationally costly, so in [6] we instead use the bound decrease resulting from just once sending messages from the candidate cluster to its intersection edges.

If we were to add the full (un-coarsened) cluster, this bound decrease would be:

$$d(c) = \sum_{ij \in c}\max_{x_i, x_j}b_{ij}(x_i, x_j) - \max_{x_c}\sum_{ij \in c}b_{ij}(x_i, x_j), \qquad (13)$$

where $b_{ij}(x_i, x_j) = \lambda_{ij \to ij}(x_i, x_j) + \sum_{c:ij \in c}\lambda_{c \to ij}(z_i^c[x_i], z_j^c[x_j])$.

Our strategy now is as follows: we add the cluster $c$ that maximizes $d(c)$, and then choose a partitioning $Z_i^c$ for all $i \in c$ that is guaranteed to achieve a decrease that is *close* to $d(c)$. This can clearly be achieved by using the trivial partition $Z_i^c = X_i$ (which achieves $d(c)$). However, in many cases it is also possible to achieve it while using much coarser partitionings.

The set of all possible partitionings $Z_i^c$ is too large to optimize over. Instead, we consider just $|X_i|$ candidate partitions that are generated based on the beliefs $b_i(x_i)$. Intuitively, the states with lower belief values $b_i(x_i)$ are less likely to influence the MAP, and can thus be bundled together. We will therefore consider partitions where the $k$ states with lowest belief values are put into the same "catch-all" coarse state $s_i^c$, and all other states of $x_i$ get their own coarse state. Formally, a partition $Z_i^c$ is characterized by a value $\kappa_i$ such that $s_i^c$ is the set of all $x_i$ with $b_i(x_i) < \kappa_i$. The question next is how big we can make the catch-all state without sacrificing the bound decrease.

We employ a greedy scheme whereby each $i \in c$ (in arbitrary order) is partitioned separately, while the other partitions are kept fixed. The process starts with $Z_i^c = X_i$ for all $i \in c$. We would like to choose $s_i^c$ such that it is sufficiently separated from the state that achieves $d(c)$. Formally, given a margin parameter $\gamma$ we choose $\kappa_i$ to be as large as possible such that the following constraint still holds[3]:

$$\max_{\substack{z_c \backslash \{z_i^c\}, \\ z_i^c = s_i^c}} \sum_{st \in c} b_{st}(z_s^c, z_t^c) \leq \max_{x_c} \sum_{st \in c} b_{st}(x_s, x_t) - \gamma,$$

where the first maximization is over the coarse variables $Z_{c \backslash i}$, and $Z_i^c$ is fixed to the catch-all state $s_i^c$ (note that the partitioning for $Z_i^c$ is a function of $\kappa_i$). We can find the optimal $\kappa_i$ in time $O(|X_i|^{|c|})$ by starting with $\kappa_i = -\infty$ and increasing it until the constraint is violated. Since each subsequent value of $s_i^c$ differs by one additional state $x_i$, we can re-use the maximizations over $\mathbf{z}_{c \backslash i}$ for the previous value of $s_i^c$ in evaluating the constraint for the current $s_i^c$.

It can be shown by induction that this results in a coarsening that has a guaranteed bound decrease of at least $d(c) + \min(0, \gamma)$. Setting $\gamma < 0$ would give a partitioning with fewer coarse states at the cost of a smaller guaranteed bound decrease. On the other hand, setting $\gamma > 0$ results in a *margin* between the value of the dual objective (after sending the coarsened cluster message) and its value if we were to fix $x_i$ in the max terms of Eq. 11 to a value in $s_i^c$. This makes it less likely that a state in $s_i^c$ will become important again in subsequent message passing iterations. For the experiments in this paper we use $\gamma = 3d(c)$, scaling $\gamma$ with the value of the guaranteed bound decrease for the full cluster. Note that this greedy algorithm does not necessarily find the partitioning with the fewest number of coarse states that achieves the bound decrease.

## 5 Experiments

We report results on the protein design problem, originally described in [9]. The protein design problem is the inverse of the protein folding problem. Given a desired backbone structure for the protein, the goal is to construct the sequence of amino-acids that results in a low energy, and thus stable, configuration. We can use an approximate energy function to guide us towards finding a set of amino-acids and rotamer configurations with minimal energy. In [9] the design problem was posed as finding a MAP configuration in a pairwise MRF. The models used there (which are also available online) have a number of states per variable that is between 2 and 158, and contain up to 180 variables per model. The models are also quite dense so that exact calculation is not feasible.

Recently we showed [6] that all but one of the problems described in [9] can be solved exactly by using a LP relaxation with clusters on three variables. However, since each individual state has roughly 100 possible values, processing triplets required $10^6$ operations, making the optimization costly. In what follows we describe two sets of experiments that show that, by coarsening, we can both significantly reduce the computation time and achieve similar performance as if we had used un-coarsened triplets [6]. The experiments differ in the strategy for adding triplets, and illustrate two performance regimes. In both experimental setups we first run the standard edge-based message passing algorithm for 1000 iterations.

In the first experiment, we add all triplets that correspond to variables whose single node beliefs are tied (within $10^{-5}$) at the maximum after running the edge-based algorithm. Since tied beliefs correspond to fractional LP solutions, it is natural to consider these in tighter relaxations. The triplets correspond to partitioned variables, as explained in Sec. 2. The partitioning is guided by the ties in the single node beliefs. Specifically, for each variable $X_i$ we find states whose single node beliefs are tied at the maximum. Denote the number of states maximizing the belief by $r$. Then, we partition

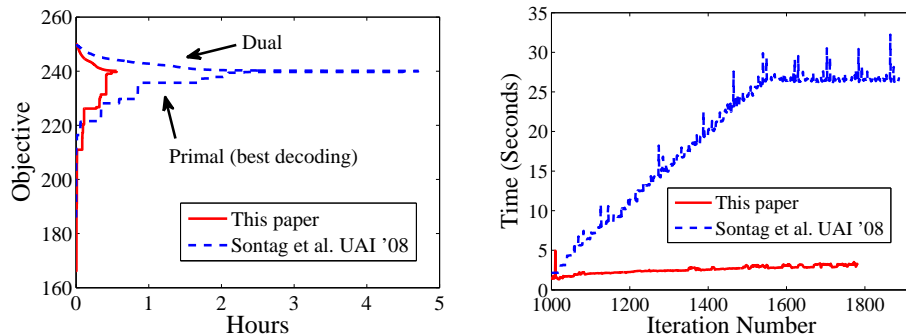

Figure 3: Comparison with algorithm from [6] for the protein "1aac", after the first 1000 iterations. **Left**: Dual objective as a function of time. **Right**: The cost per one iteration over the entire graph.

the states into $r$ subsets, each containing a different maximizing state. The other (non-maximizing) states are split randomly among the $r$ subsets. The triplets are then constructed over the coarsened variables $Z_i^c$ and the message passing algorithm of Sec. 3 is applied to the resulting structure. After convergence of the algorithm, we recalculate the single node beliefs. These may result in a different partition scheme, and hence new variables $Z_i^c$. We add new triplets corresponding to the new variables and re-run. We repeat until the dual-LP bound is sufficiently close to the value of the integral assignment obtained from the messages (note that these values would not coincide if the relaxation were not tight; in these experiments they do, so the final relaxation is tight).

We applied the above scheme to the ten smallest proteins in the dataset used in [6] (for the larger proteins we used a different strategy described next). We were able to solve all ten exactly, as in [6]. The mean running time was six minutes. The gain in computational efficiency as a result of using coarsened-triplets was considerable: The average state space size for coarsened triplets was on average 3000 times smaller than that of the original triplet state space, resulting in a factor 3000 speed gain over a scheme that uses the complete (un-coarsened) triplets.[4] This big factor comes about because a very small number of states are tied per variable, thus increasing the efficiency of our method where the number of partitions is equal to the number of tied states. While running on full triplets was completely impractical, the coarsened message passing algorithm is very practical and achieves the exact MAP assignments.

Our second set of experiments follows the setup of [6] (see Sec. 3), alternating between adding 5 triplets to the relaxation and running MPLP for 20 more iterations. The only difference is that, after deciding to add a cluster, we use the algorithm from Sec. 4 to partition the variables. We tried various settings of $\gamma$, including $\gamma = 0$ and .01, and found that $\gamma = 3d(c)$ gave the best overall runtimes.

We applied this second scheme to the 15 largest proteins in the dataset.[5] Of these, we found the exact MAP in 47% of the cases (according to the criterion used in [6]), and in the rest of the cases were within $10^{-2}$ of the known optimal value. For the cases that were solved exactly, the mean running time was 1.5 hours, and on average the proteins were solved 8.1 times faster than with [6].[6] To compare the running times on all 15 proteins, we checked how long it took for the difference between the dual and primal objectives to be less than $.01f(\boldsymbol{x}_M; \boldsymbol{\theta})$, where $\boldsymbol{x}_M$ is the MAP assignment. This revealed that our method is faster by an average factor of 4.3. The reason why these factors are less than the 3000 in the previous setup is that, for the larger proteins, the number of tied states is typically much higher than that for the small ones.

Results for one of the proteins that we solved exactly are shown in Fig. 3. The cost per iteration increases very little after adding each triplet, showing that our algorithm significantly coarsened the clusters. The total number of iterations and number of triplets added were roughly the same. Two triplet clusters were added twice using different coarsenings, but otherwise each triplet only needed to be added once, demonstrating that our algorithm chose the right coarsenings.

# 6  Discussion

We presented an algorithm that enforces higher-order consistency constraints on LP relaxations, but at a reduced computational cost. Our technique further explores the trade-offs of representing complex constraints on the marginal polytope while keeping the optimization tractable. In applying the method, we chose to cluster variables' states based a bound minimization criterion after solving using a looser constraint on the polytope.

A class of approaches related to ours are the "coarse-to-fine" applications of belief propagation [1, 4]. In those, one solves low-resolution versions of an MRF, and uses the resulting beliefs to initialize finer resolution versions. Although they share the element of coarsening with our approach, the goal of coarse-to-fine approaches is very different from our objective. Specifically, the low-resolution MRFs only serve to speed-up convergence of the full resolution MRF via better initialization. Thus, one typically should not expect it to perform better than the finest granularity MRF. In contrast, our approach is designed to strictly improve the performance of the original MRF by introducing additional (coarse) clusters. One of the key technical differences is that in our formulation the setting of coarse and fine variables are refined iteratively whereas in [1], once a coarse MRF has been solved, it is not revisited.

There are a number of interesting directions to explore. Using the same ideas as in this paper, one can introduce coarsened pairwise consistency constraints in addition the full pairwise consistency constraints. Although this would not tighten the relaxation, by passing messages more frequently in the coarsened space, and only occasionally revisiting the full edges, this could give significant computational benefits when the nodes have large numbers of states. This would be much more similar to the coarse-to-fine approach described above.

With the coarsening strategy used here, the number of variables still grows exponentially with the cluster size, albeit at a lower rate. One way to avoid the exponential growth is to partition the states of a cluster into a fixed number of states (e.g., two), and then constrain such partitions to be consistent with each other. Such a process may be repeated recursively, generating a hierarchy of coarsened variables. The key advantage in this approach is that it represents progressively larger clusters, but with no exponential growth. An interesting open question is to understand how these hierarchies should be constructed.

Our techniques may also be helpful for finding the MAP assignment in MRFs with structured potentials, such as context-specific Bayesian networks. Finally, these constraints can also be used when calculating marginals.

## Footnotes

[1]We do not use potentials on single nodes $\theta_i(x_i)$ since these can be *folded* into $\theta_{ij}(x_i, x_j)$. Our algorithm can also be derived with explicit $\theta_i(x_i)$, and we omit the details for brevity.

[2]We use a superscript of $c$ to highlight the fact that different clusters may use different partitionings for $Z_i$. Also, there can be multiple clusters on the same set of variables, each using a different partitioning.

[3]The constraint may be infeasible for $\gamma > 0$, in which case we simply choose $Z_i^c = X_i$.

[4]These timing comparisons do not apply to [6] since that algorithm did not use all the triplets.

[5]We do not run on the protein 1fpo, which was not solved in [6].

[6]We made sure that differences were not due to different processing powers or CPU loads.

# References

[1] P. F. Felzenszwalb and D. P. Huttenlocher. Efficient belief propagation for early vision. *Int. J. Comput. Vision*, 70(1):41–54, 2006.

[2] A. Globerson and T. Jaakkola. Fixing max-product: Convergent message passing algorithms for MAP LP-relaxations. In *Advances in Neural Information Processing Systems 21*. MIT Press, 2008.

[3] V. Kolmogorov. Convergent tree-reweighted message passing for energy minimization. *IEEE Trans. Pattern Anal. Mach. Intell.*, 28(10):1568–1583, 2006.

[4] C. Raphael. Coarse-to-fine dynamic programming. *IEEE Transactions on Pattern Analysis and Machine Intelligence*, 23(12):1379–1390, 2001.

[5] D. Sontag and T. Jaakkola. New outer bounds on the marginal polytope. In *Advances in Neural Information Processing Systems 21*. MIT Press, 2008.

[6] D. Sontag, T. Meltzer, A. Globerson, Y. Weiss, and T. Jaakkola. Tightening LP relaxations for MAP using message-passing. In *UAI*, 2008.

[7] M. Wainwright and M. I. Jordan. Graphical models, exponential families and variational inference. Technical report, UC Berkeley, Dept. of Statistics, 2003.

[8] M. Wainwright and M. I. Jordan. Log-determinant relaxation for approximate inference in discrete Markov random fields. *IEEE Transactions on Signal Processing*, 54(6):2099–2109, June 2006.

[9] C. Yanover, T. Meltzer, and Y. Weiss. Linear programming relaxations and belief propagation – an empirical study. *JMLR*, 7:1887–1907, 2006.

[10] J.S. Yedidia, W.T. Freeman, and Y. Weiss. Constructing free-energy approximations and generalized belief propagation algorithms. *IEEE Trans. on Information Theory*, 51(7):2282– 2312, 2005.
